# Stability and Observability

Max Garzon

Fernanda Botelho

garzonm@hermes.msci.memst.edu  botelhof@hermes.msci.memst.edu

Institute for Intelligent Systems  Department of Mathematical Sciences

Memphis State University

Memphis, TN 38152  U.S.A.

The theme was the effect of perturbations of the defining parameters of a neural network due to: 1) *measurements* (particularly with analog networks); 2) *discretization* due to *a*) digital implementation of analog nets; *b*) bounded-precision implementation of digital networks; or *c*) inaccurate evaluation of the transfer function(s); 3) *noise* in or incomplete input and/or output of the net or individual cells (particularly with analog networks).

The workshop presentations address these problems in various ways. Some develop models to understand the influence of errors/perturbation in the output, learning and general behavior of the net (probabilistic in Piche and Tresp; optimization in Rojas; dynamical systems in Botelho & Garzon). Others attempt to identify desirable properties that are to be preserved by neural network solutions (equilibria under faster convergence in Peterfreund & Baram; decision regions in Cohen). Of particular interest is to develop networks that compute robustly, in the sense that small perturbations of their parameters do not affect their dynamical and observable behavior (stability in biological networks in Chauvet & Chauvet; oscillation stability in learning in Rojas; hysterectic finite-state machine simulation in Casey). In particular, understand how biological networks cope with uncertainty and errors (Chauvet & Chauvet) through the type of stability that they exhibit.

## QUESTIONS AND ANSWERS

Some questions served to focus the presentations and discussion. Some were (partially) answered, and others were barely touched:

◇ *What are the most significant errors in defining parameters with respect to output behavior?* By evidence presented, i/o and weights seem to be the most sensitive.

◇ *Is there an essential difference between perturbations in weights (long-term memory) and inputs (short-memory)?* They seem to play a symmetric role in feedforward and, to some extent, recurrent nets. But evidence is not conclusive.

◇ *How can the effects of perturbations be kept under control or eliminated altogether?* If one is only interested in dynamical qualitative features, small enough errors of any kind (as incurred in digital implementations for example) are not relevant for most nets (What you see on the screen is what should be happening).

◇ *Are they architecture (in)dependent?* On the other hand, they spread rapidly under iteration and exact quantification varies with the architecture.

◇ *Are stability and implementation based on dynamical features the only ways to*

*cope with errors/perturbations?* The difficulty to quantify (perhaps due to lack of research) seems to indicate so. Stability worth a closer look for its own sake.

◊ *Does requiring robust computation really restrict the capabilities of neural networks?* Apparently not, since in all likelihood there exist universal neural nets which tolerate small errors (see talk by Botelho & Garson). Wide open.

## TALKS AND SHORT ABSTRACTS

• **Trajectory Control of Convergent Networks**, *Natan Peterfreund and Y. Baram.* We present a class of feedback control functions which accelerate convergence rates of autonomous nonlinear dynamical systems such as neural network models, without affecting the basic convergence properties (e.g. equilibrium points). natan@tx.technion.ac.il

• **Sensitivity of Neural Network to Errors**, *Steven Piche.* Using stochastic models, analytic expressions for the effects of such errors are derived for arbitrary feedforward neural networks. Both, the degree of nonlinearity and the relationship between input correlation and the weight vectors, are found to be important in determining the effects of errors. piche@mcc.com

• **Stability of Learning in Neural Networks**, *Raul Rojas.* Finding optimal combinations of learning and momentum rates for the standard backpropagation involves difficult tradeoffs across fractal boundaries. We show that statistic preprocessing can bring error functions under control. rojas@inf.fu-berlin.de

• **Stability of Purkinje Cells in Cerebellar Cortex**, *Gilbert Chauvet and Pierre Chauvet.* The cerebellar cortex (involved in learning and retrieving) is a *hierarchical* functional unit built around a Purkinje cell, which has its own functional properties. We have shown experimentally that Purkinje dynamical systems have a unique solution, which is asymptotically stable. It seems possible to give a general explanation of stability in biological systems. chauvet@ibt.univ-angers.fr.

• **Recall and Learning with Deficient Data**, *Volker Tresp, Subutai Ahmad, Ralph Neuneier.* Mean values and maximum likelihood estimators are not the best ways to cope with noisy data. See their LA:5 poster summary in these proceedings for an extended abstract. tresp@zfe.siemens.de

• **Computation Dynamics in Discrete-Time Recurrent Networks**, *Mike Casey.* We consider training recurrent higher-order neural networks to recognize regular languages, using the cycles in their diagrams for hysterectic simulation of finite state machines. The latter suggests a general logical approach to solving the 'neural code' problem for living organisms, necessary for understanding information processing in the nervous system. mcasey@sdcc.ucsd.edu

• **Synthesis of Decision Regions in Dynamical Systems**, *Mike Cohen.* As a first step toward a representation theory of decision functions via neural nets, he presented a method which enables the construction of a system of differential equations exhibiting a given finite set of decision regions and equilibria with a very large class of indices consistent with the Morse inequalities. mike@park.bu.edu

• **Observability of Discrete and Analog Networks**, *F. Botelho and M. Garzon.* We show that most networks (with finitely many analog or infinitely many boolean neurons) are *observable* (i.e., all their corrupted pseudo-orbits actually reflect true orbits). See their DS:2 poster summary in these proceedings.